# Learning Nonparametric Models for Probabilistic Imitation

**David B. Grimes**      **Daniel R. Rashid**      **Rajesh P.N. Rao**

Department of Computer Science
University of Washington
Seattle, WA 98195
`grimes,rashid8,rao@cs.washington.edu`

## Abstract

Learning by imitation represents an important mechanism for rapid acquisition of new behaviors in humans and robots. A critical requirement for learning by imitation is the ability to handle uncertainty arising from the observation process as well as the imitator's own dynamics and interactions with the environment. In this paper, we present a new probabilistic method for inferring imitative actions that takes into account both the observations of the teacher as well as the imitator's dynamics. Our key contribution is a nonparametric learning method which generalizes to systems with very different dynamics. Rather than relying on a known forward model of the dynamics, our approach learns a nonparametric forward model via exploration. Leveraging advances in approximate inference in graphical models, we show how the learned forward model can be directly used to plan an imitating sequence. We provide experimental results for two systems: a biomechanical model of the human arm and a 25-degrees-of-freedom humanoid robot. We demonstrate that the proposed method can be used to learn appropriate motor inputs to the model arm which imitates the desired movements. A second set of results demonstrates dynamically stable full-body imitation of a human teacher by the humanoid robot.

## 1   Introduction

A fundamental and versatile mechanism for learning in humans is imitation. Infants as young as 42 minutes of age have been found to imitate facial acts such as tongue protrusion while older children can perform complicated forms of imitation ranging from learning to manipulate novel objects in particular ways to imitation based on inference of goals from unsuccessful demonstrations (see [11] for a review). Robotics researchers have become increasingly interested in learning by imitation (also called "learning by watching" or "learning from demonstration") as an attractive alternative to manually programming robots [5, 8, 19]. However, most of these approaches do not take uncertainty into account. Uncertainty in imitation arises from many sources including the internal dynamics of the robot, the robot's interactions with its environment, observations of the teacher, etc. Being able to handle uncertainty is especially critical in robotic imitation because executing actions that have high uncertainty during imitation could lead to potentially disastrous consequences.

In this paper, we propose a new technique for imitation that explicitly handles uncertainty using a probabilistic model of actions and their sensory consequences. Rather than relying on a physics-based parametric model of system dynamics as in traditional methods, our approach learns a nonparametric model of the imitator's internal dynamics during a constrained exploration period. The learned model is then used to infer appropriate actions for imitation using probabilistic inference in a dynamic Bayesian network (DBN) with teacher observations as evidence. We demonstrate the viability of the approach using two systems: a biomechanical model of the human arm and a 25-

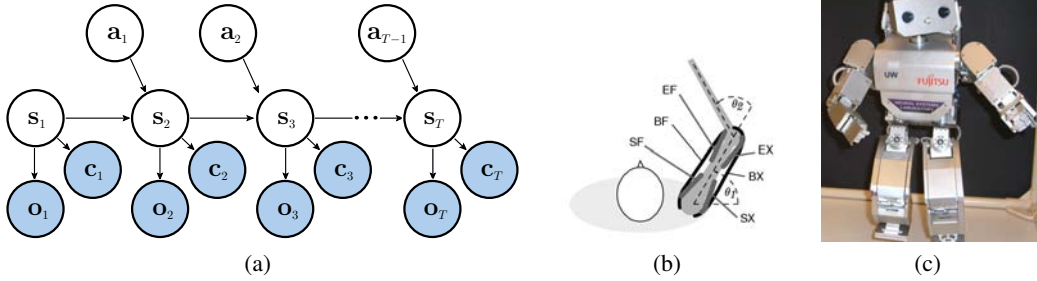

(a)                         (b)             (c)

Figure 1: **Graphical model and systems for imitation learning.** (a) Dynamic Bayesian network for inferring a sequence of imitative actions $\mathbf{a}_{1:T-1}$ from a sequence of observations of the teacher $\mathbf{o}_{1:T}$. The model also allows for probabilistic constraint variables $\mathbf{c}_t$ on the imitators states $\mathbf{s}_t$. Nonparametric model learning constructs the model $P(\mathbf{s}_{t+1}|\mathbf{s}_t, \mathbf{a}_t)$ from empirical data. (b) The two link biomechanical model of the human arm (from [10]) used in experiments on learning reaching movements via imitation. (c) The Fujitsu Hoap-2 humanoid robot used in our experiments on full-body, dynamic imitation.

degrees-of-freedom humanoid robot. Our first set of results illustrate how the proposed method can be used to learn appropriate motor commands for producing imitative movements in the model human arm. The second set of results demonstrates dynamically stable full-body imitation of a human teacher by the humanoid robot. Taken together, the results suggest that a probabilistic approach to imitation based on nonparametric model learning could provide a powerful and flexible platform for acquiring new behaviors in complex robotic systems.

## 2 Imitation via Inference and Constrained Exploration

In this section we present our inference-based approach to selecting a set of actions based on observations of another agent's state during demonstration, and a set of probabilistic constraints. We present our algorithms within the framework of the graphical model shown in Fig. 1(a). We denote the sequence of continuous action variables $\mathbf{a}_1, \cdots, \mathbf{a}_t, \cdots, \mathbf{a}_{T-1}$. We use the convention that the agent starts in an initial state $\mathbf{s}_1$, and as the result of executing the actions visits the set of continuous states $\mathbf{s}_2, \cdots, \mathbf{s}_t, \cdots, \mathbf{s}_T$. Note that an initial action $\mathbf{a}_0$ can be trivially included.

In our imitation learning framework the agent observes a sequence of continuous variables $\mathbf{o}_1, \cdots, \mathbf{o}_t, \cdots, \mathbf{o}_{T-1}$ providing partial information about the state of the teacher during demonstration. The conditional probability density $P(\mathbf{o}_t|\mathbf{s}_t)$ encodes how likely an observation of the teacher ($\mathbf{o}_t$) agrees with an an agent's state ($\mathbf{s}_t$) while performing the same motion or task. This marks a key difference with the Partially Observable Markov Decision Process (POMDP) framework. Here the observations are of the demonstrator (generally with different embodiment), and we currently assume that the learner can observe it's own state.

Probabilistic constraints on state variables are included within the graphical model by a set of variables $\mathbf{c}_t$. The corresponding constraint models $P(\mathbf{c}_t|\mathbf{s}_t)$ encode the likelihood of satisfying the constraint in state $\mathbf{s}_t$. Constraint variables are used in our framework to represent goals such as reaching a desired goal state ($\mathbf{c}_T = \mathbf{s}_G$), or a going through a way point ($\mathbf{c}_t = \mathbf{s}_W$). The choice of the constraint model is domain dependent. Here we utilize a central Gaussian density $P(\mathbf{c}_t|\mathbf{s}_t) = \mathcal{N}(\mathbf{c}_t - \mathbf{s}_t; 0, \Sigma_c)$. The variance parameter for each constraint may be set by hand using domain knowledge, or could be learned using feedback from the environment.

Given a set of evidence $\mathcal{E} \subseteq \{\mathbf{o}_1, \cdots, \mathbf{o}_T, \mathbf{c}_1, \cdots, \mathbf{c}_T\}$ we desire actions which maximize the likelihood of the evidence. Although space limitations rule out a thorough discussion, to achieve a tractable inference we focus here on computing marginal posteriors over each action rather than the *maximum a posteriori* (MAP) sequence. While in principle any algorithm for computing the marginal posterior distributions of the action variables could be used, we find it convenient here to use Pearl's belief propagation (BP) algorithm [13]. BP was originally restricted to tree structured graphical models with discrete variables. Several advances have broadened the applicability to general graph struc-

tures [18] and to continuous variables in undirected graph structures [16]. Here we derive belief propagation for the directed case though we note that the difference is largely a semantic convenience, as any Bayesian network can be represented as a Markov random field, or more generally a factor graph [9]. Our approach is most similar to Nonparametric Belief Propagation (NBP) [16], with key differences highlighted throughout this section.

The result of performing belief propagation is a set of marginal belief distributions $B(\mathbf{x}) = P(\mathbf{x}|\mathcal{E}) = \pi(\mathbf{x})\lambda(\mathbf{x})$. This belief distribution is the product of two sets of messages $\pi(\mathbf{x})$ and $\lambda(\mathbf{x})$, which represent the information coming from neighboring parent and children variable nodes respectively. Beliefs are computed via messages passed along the edges of the graphical model, which are distributions over single variables. The $i$-th parent of variable $\mathbf{x}$ passes to $\mathbf{x}$ the distribution $\pi_{\mathbf{x}}(\mathbf{u}_i)$. Child $j$ of variable $\mathbf{x}$ would pass to $\mathbf{x}$ the distribution $\lambda_{\mathbf{y}_j}(\mathbf{x})$. In the discrete (finite space) case, messages are easily represented by discrete distributions. For the case of arbitrary continuous densities message representation is in itself a challenge. As we propose a nonparametric, model-free approach to learning system dynamics it follows that we also want to allow for (approximately) representing the multi-modal, non-Gaussian distributions that arise during inference. As in the NBP approach [16] we adopt the use of a mixture of Gaussian kernels (Eq. 5) to represent arbitrary message and belief distributions.

For convenience we treat observed and hidden variables in the graph identically by allowing a node $X$ to send itself the message $\lambda_X(\mathbf{x})$. This "self message" represented using a Dirac delta distribution about the observed value is considered in the product of all messages from the $m$ children (denoted $Y_j$) of $X$:

$$\lambda(\mathbf{x}) = \lambda_X(\mathbf{x}) \prod_j^m \lambda_{Y_j}(\mathbf{x}).  \qquad (1)$$

Messages from parent variables are incorporated by integrating the conditional probability of $\mathbf{x}$ over all possible values of the $k$ parents times the probability that combination of values as evaluated in the corresponding messages from a parent node:

$$\pi(\mathbf{x}) = \int_{\mathbf{u}_{1:n}} P(\mathbf{x}|\mathbf{u}_1, \cdots, \mathbf{u}_n) \prod_i^n \pi_X(\mathbf{u}_i) \, d\mathbf{u}_{1:n}.  \qquad (2)$$

Messages are updated according to the following two equations:

$$\lambda_X(\mathbf{u}_j) = \int_{\mathbf{x}} \lambda(\mathbf{x}) \int_{\mathbf{u}_{1:n/j}} P(\mathbf{x}|\mathbf{u}_1, \cdots, \mathbf{u}_n) \prod_{i \neq j}^n \pi_{\mathbf{x}}(\mathbf{u}_i) \, d\mathbf{u}_{1:n/j} d\mathbf{x}  \qquad (3)$$

$$\pi_{Y_j}(\mathbf{x}) = \pi(\mathbf{x})\lambda_X(\mathbf{x}) \prod_{i \neq j} \lambda_{Y_i}(\mathbf{x}).  \qquad (4)$$

The main operations in Eqs 1-4 are integration and multiplication of mixtures of Gaussians. The evaluation of integrals will be discussed after first introducing Gaussian Mixture Regression in Sec. 3. Although the product of a set of mixtures of Gaussians is simply another mixture of Gaussians, the complexity (in terms of the number of components in the output mixture) grows exponentially in the number of input mixtures. Thus an approximation is needed to keep inference tractable in the action sequence length $T$. Rather than use a multiscale sampling method to obtain a set of representative particles from the product as in [7] we first assume that we can compute the exact product density for a given set of input mixtures. We then apply the simple heuristic of keeping a fixed number of mixture components, which through experimentation we found to be highly effective. This heuristic is based on the empirical sparseness of product mixture components' prior probabilities. For example, when the message $\pi_{\mathbf{s}_{t+1}}(\mathbf{s}_t)$ coming from a previous state has $M = 10$ components, the message from the action $\pi_{\mathbf{a}_{t-1}}(\mathbf{a}_t)$ has $N = 1$ component (based on a unimodal Gaussian prior), and the GMR model has $P = 67$ components, the conditional product has $MNP = 670$ components. However, we see experimentally that less than ten components have a weight which is within five orders of magnitude of the maximal weight. Thus we can simply select the top $K' = 10$ components. This sparsity should not be surprising as the $P$ model components represent localized data, and only a few of these components tend to have overlap with the belief state being propagated. Currently we fix $K'$ although an adaptive mechanism could further speed inference.

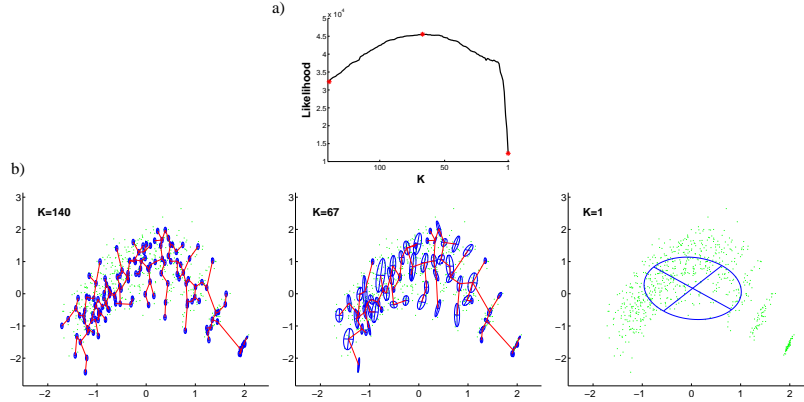

Figure 2: **Nonparametric GMR model selection.** a) The value of our model selection criteria rises from the initial model with $K = L$ components, to a peak around 67 components, after which it falls off. b) The three series of plots show the current parameters of the model (blue ellipses), layered over the set of regression test points (in green), and the minimum spanning tree (red lines) between neighboring components. Shown here is a projection of the 14-dimensional data (projected onto the first two principal components of the training data).

We now briefly describe our algorithm [1] for action inference and constrained exploration. The inputs to the action inference algorithm are the set of evidence $\mathcal{E}$, and an instance of the dynamic Bayesian network $\mathcal{M} = \{P_S, P_A, P_F, P_O, P_C\}$, composed of the prior on the initial state, the prior on actions, the forward model, the imitation observation model, and the probabilistic constraint model respectively. Inference proceeds by first "inverting" the evidence from the observations and constraint variables yielding the messages $\lambda_{\mathbf{o}_t}(\mathbf{s}_t)$, $\lambda_{\mathbf{c}_t}(\mathbf{s}_t)$. After initialization from the priors $P_S$, $P_A$ we perform a forward planning pass thereby computing the forward state messages $\pi_{\mathbf{s}_{t+1}}(\mathbf{s}_t)$. Similarly a backward planning sweep produces the messages $\lambda_{\mathbf{s}_t}(\mathbf{s}_{t-1})$. The algorithm then combines information from forward and backward messages (via Eq. 3) to compute belief distributions of actions. We then select the maximum marginal belief action $\hat{\mathbf{a}}_t$ from the belief distribution using the mode finding algorithm described in [4].

Our algorithm to iteratively explore the state and action spaces while satisfying the constraints placed on the system builds on the inference based action selection algorithm described above. The inputs are an initial model $\mathcal{M}_0$, a set of evidence $\mathcal{E}$, and a number of iterations to be performed $N$. At each iteration we infer a sequence of maximum marginal actions and execute them. Execution yields a sequence of states, which are used to update the learned forward model (see Section 3). Using the new (ideally more accurate) forward model we show we are able to obtain a better imitation of the teacher via the newly inferred actions. The final model and sequence of actions are then returned after $N$ constrained exploration trials.

For simplicity, we currently assume that state and action prior distributions, and the observation and constraint models are pre-specified. Evidence from our experiments shows that specifying these parts of the model is not overly cumbersome even in the real-world domains we have studied. The focus of our algorithms presented here is to learn the forward model which in many real-world domains is extremely complex to derive analytically. Sections 4.1 and 4.2 describe the results of our algorithms applied in the human arm model and humanoid robot domains respectively.

## 3 Nonparametric Model Learning

In this section we investigate an algorithm for learning a nonparametric forward model via Gaussian Mixture Regression (GMR) [17]. The motivation behind selecting GMR is that it allows for closed form evaluation of the integrals found in Eqs 1-4. Thus it allows efficient inference without the need to resort to Monte Carlo (sample based) approximations in inference.

The common Gaussian Mixture Model (GMM) forms the basis of Gaussian Mixture Regression:

$$p(\mathbf{x}|\theta) = \sum_k p(k|\theta_k)p(\mathbf{x}|k,\theta_k) = \sum_k w_k \mathcal{N}(\mathbf{x};\mu_k,\Sigma_k). \tag{5}$$

We now assume that the random variable $X$ is formed via the concatenation of the $n$ random variables $X_1, X_2, \cdots, X_n$, such that $\mathbf{x} = [\mathbf{x}_1^\top \mathbf{x}_2^\top \cdots \mathbf{x}_n^\top]^\top$. The theorem of Gaussian conditioning states that if $\mathbf{x} \sim \mathcal{N}(\mu,\Sigma)$ where $\mu = [(\mu_i)]$ and $\Sigma = [(\Sigma_{ij})]$ then the variable $X_i$ is normally distributed given $X_j$:

$$p(X_i = \mathbf{x}_i | X_j = \mathbf{x}_j) = \mathcal{N}(\mu_i + \Sigma_{ij}\Sigma_{jj}^{-1}(\mathbf{x}_j - \mu_j), \Sigma_{ii} - \Sigma_{ij}\Sigma_{jj}^{-1}\Sigma_{ji}). \tag{6}$$

Gaussian mixture regression is derived by applying the result of this theorem to Eq. 5:

$$p(\mathbf{x}_i|\mathbf{x}_j,\theta) = \sum_k w_{kj}(\mathbf{x}_j)\mathcal{N}(\mathbf{x}_i;\mu_{kij}(\mathbf{x}_j),\Sigma_{kij}). \tag{7}$$

We use $\mu_{kj}$ denote the mean of the $j$-th variable in the $k$-th component of the mixture model. Likewise $\Sigma_{kij}$ denotes the covariance between the variables $\mathbf{x}_i$ and $\mathbf{x}_j$ in the $k$-th component. Instead of a fixed weight and mean for each component we now have a weight function dependent on the conditioning variable $\mathbf{x}_j$:

$$w_{kj}(\mathbf{x}) = \frac{w_k \mathcal{N}(\mathbf{x};\mu_{kj},\Sigma_{kjj})}{\sum_{k'} w_{k'} \mathcal{N}(\mathbf{x};\mu_{k'j},\Sigma_{k'jj})}. \tag{8}$$

Likewise the mean of the $k$-th conditioned component of $\mathbf{x}_i$ given $\mathbf{x}_j$ is a function of $\mathbf{x}_j$:

$$\mu_{kij}(\mathbf{x}) = \mu_{ki} + \Sigma_{kij}\Sigma_{kjj}^{-1}(\mathbf{x} - \mu_{kj}). \tag{9}$$

Belief propagation requires the evaluation of integrals convolving the conditional distribution of one variable $\mathbf{x}_i$ given a GMM distribution $\gamma(\cdot,\theta')$ of another variable $\mathbf{x}_j$:

$$\int p(\mathbf{x}_i|\mathbf{x}_j,\theta)\gamma(\mathbf{x}_j;\theta')d\mathbf{x}_j \tag{10}$$

Fortunately rearranging the terms in the densities reduces the product of the two GMMs to a third GMM, which is then marginalized w.r.t. $\mathbf{x}_i$ under the integral operator.

We now turn to the problem of learning a GMR model from data. As the learning methodology we wish to adopt is non-parametric we do not want to select the number of components $K$ *a priori*. This rules out the common strategy of using the well known expectation maximization (EM) algorithm for learning a model of the full joint density $p(\mathbf{x})$. Although Bayesian strategies exist for selecting the number of components, as pointed out by [17] a joint density modeling approach rarely yields the best model under a regression loss function. Thus we adopt a very similar algorithm to the Iterative Pairwise Replace Algorithm (IPRA) [15, 17] which simultaneously performs model fitting and selection of the GMR model parameters $\theta$.

We assume that a set of state and action histories have been observed during the $N$ trial histories: $\{[\mathbf{s}_1^i, \mathbf{a}_1^i, \mathbf{s}_2^i, \mathbf{a}_2^i, \cdots, \mathbf{a}_{T-1}^i \mathbf{s}_T^i]\}_{i=1}^N$. To learn a GMR forward model we first construct the joint variable space: $\mathbf{x} = [\mathbf{s}^\top \mathbf{a}^\top (\mathbf{s}')^\top]^\top$ where $\mathbf{s}'$ denotes the resulting state when executing action $\mathbf{a}$ in state $\mathbf{s}$. The time-invariant dataset is then represented with by matrix $X_{tr} = [\mathbf{x}^1, \cdots, \mathbf{x}^L]$

Model learning and selection first constructs the fully non-parametric representation of the training set with $K = L$ isotropic mixture components centered on each data point $\mu_k = \mathbf{x}^k$. This parametrization is exact at making predictions at points within the training set, but generalizes extremely poorly. The algorithm proceeds by merging components which are very similar, as determined by a symmetric similarity metric between two mixture components. Following [17] we use the Hellinger distance metric. To perform efficient merging we first compute the minimum spanning tree of all mixture components. Iteratively, the algorithm merges the closest pair in the minimum spanning tree. Merging continues until there is only a single Gaussian component left. Merging the two mixtures requires computation of new local mixture parameters (to fit the data covered by both). Rather than the "method of moments" (MoM) approach to merging components and then later running expectation maximization to fine-tune the selected model, we that found performing

local maximum likelihood estimation (MLE) within model selection to be more effective at finding an accurate model.

In order to effectively perform MLE merges we first randomly partition the training data into two sets: one of "basis" vectors that we compute the minimum spanning tree on, and one of regression data points. In our experiments here we used a random fifth of the data for basis vectors. The goal of our modified IPRA algorithm is to find the model which best describes the regression points. We then define the regression likelihood over the current GMR model parameters $\theta$:

$$\mathcal{L}(\theta, X_{tr}) = \sum_{l}^{L} \sum_{i}^{n} p(\mathbf{x}_i^l | \mathbf{x}_{1,2,i-1,i+1,n}^l, \theta). \tag{11}$$

The model of size $K$ which maximizes this criteria is returned for use in our inference procedure. Fig. 2 demonstrates the learning of a forward model for the biomechanical arm model from Section 4.1. We found the regression-based model selection criteria to be effective at generalizing well outside both the basis and regression sets.

# 4  Results

## 4.1  Imitating Reaching Movements

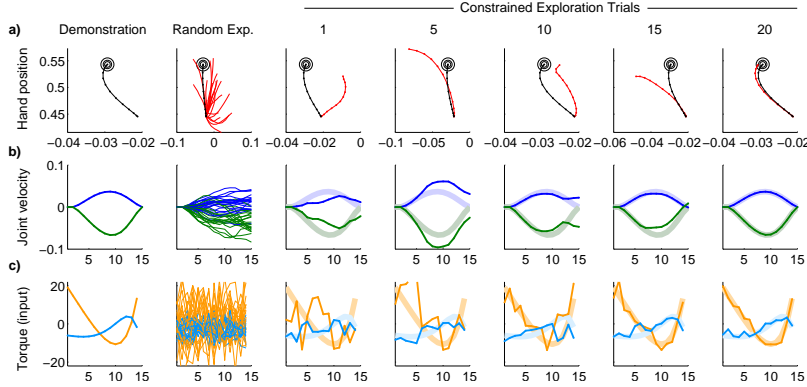

Figure 3: **Learning to imitate a reaching motion.** a) The first row shows the motion of the teacher's hand (shown in black) in Cartesian space, along with the target position. The imitator explores the space via body babbling (second plot, shown in red). From this data a GMR model is learned, constrained exploration is performed to find an imitative reaching motion (shown every 5 iterations). b) The velocities of the two joints during the imitation learning process. By trial number 20 the imitator's velocities (thin lines) closely match the demonstrator's velocities (the thick,light colored lines), and meet the zero final velocity constraint. c) The teacher's original muscle torques, followed by the babbling torques, and the torques computed during constrained exploration.

In the first set of experiments we learn reaching movements via imitation in a complex non-linear model of the human arm. The arm simulator we use is a biomechanical arm model [10] consisting of two degrees of freedom (denoted $\theta$) representing the shoulder and elbow joints. The arm is controlled via two torque inputs (denoted $\tau$) for the two degrees of freedom. The dynamics of the arm are described via the following differential equation:

$$M(\theta)\ddot{\theta} + C(\theta, \dot{\theta}) + B(\dot{\theta}) = \tau \tag{12}$$

where $M$ is the inertial force matrix, $C$ is a vector of centripetal and Coriolis forces, and $B$ is the matrix of force due to friction at the joints.

Fig. 3 shows the process of learning to perform a reaching motion via imitation. First we compute the teacher's simulated arm motion using the model-based iLQG algorithm [10] based on start and target positions of the hand. By executing the sequence of computed torque inputs $[\hat{\mathbf{a}}_{1:T-1}]$ from a specified initial state $\mathbf{s}_1$, we obtain the state history of the demonstrator $[\hat{\mathbf{s}}_{1:T}]$. To simulate the

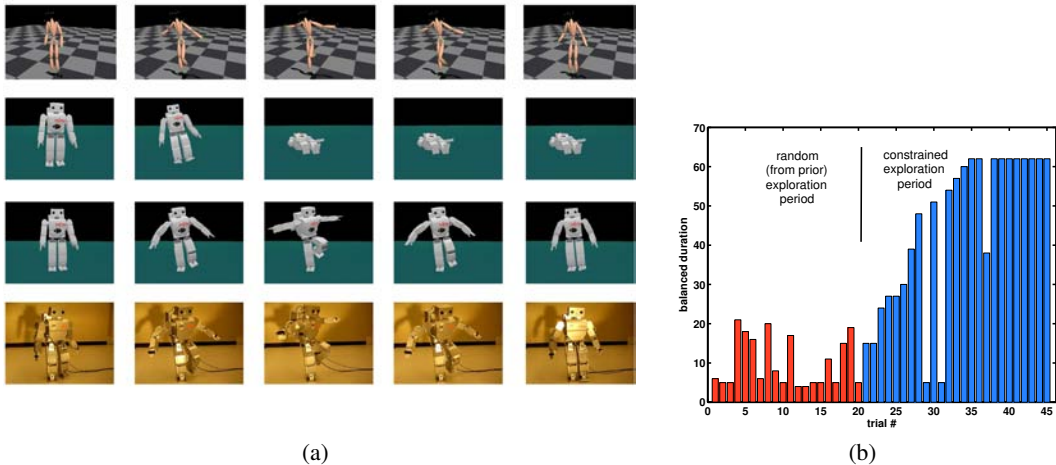

<div align="center">(a)&emsp;&emsp;&emsp;&emsp;&emsp;&emsp;&emsp;&emsp;&emsp;&emsp;&emsp;&emsp;&emsp;&emsp;&emsp;(b)</div>

Figure 4: **Humanoid robot dynamic imitation.** a) The first row consists of frames from an IK fit to the marker data during observation. The second row shows the result of performing a kinematic imitation in the simulator. The third and fourth rows show the final imitation result obtained by our method of constrained exploration, in the simulator, and on the Hoap-2 robot. b) The duration that the executed imitation was balanced (out of a total of $T = 63$) shown vs the trial number. The random exploration trials are shown in red, and the inferred imitative trials are shown in blue. Note that the balanced duration rapidly rises and by the 15th inferred sequence is able to perform the imitation without falling.

natural partial observability of a human demonstrator and a human learner, we provide our inference algorithm with noisy measurements of the kinematic state only (not the torques). A probabilistic constraint dictates that the final joint velocities be very close to zero.

## 4.2 Dynamic Humanoid Imitation

We applied our algorithms for nonparametric action selection, model learning, and constrained exploration to the problem of full-body dynamic imitation in a humanoid robot. The experiment consisted of a humanoid demonstrator performing motions such as squatting and standing on one leg. Due to space limitations only briefly describe the experiments, for more details see [6].

First, the demonstrators' kinematics were obtained using a commercially available retroreflective marker-based optical motion capture system based on inverse kinematics (IK). The IK skeletal model of the human was restricted to have the same degrees of freedom as the Fujitsu Hoap-2 humanoid robot.

Representing humanoid motion using a full kinematic configuration is problematic (due to the curse of dimensionality). Fortunately with respect to a wide class of motion (such as walking, kicking, squatting) the full number of degrees of freedom (25 in the Hoap-2) is highly redundant. For simplicity here, we use linear principal components analysis (PCA) but we are investigating the use of non-linear embedding techniques. Using PCA we were able to represent the observed instructor's kinematics in a compact four-dimensional space, thus forming the first four dimensions of the state space.

The goal of the experiment is to perform dynamic imitation, i.e. considering the dynamic balance involved in stably imitating the human demonstrator. Dynamic balance is considered using a sensor-based model. The Hoap-2 robot's sensors provide measurements of the angular rotation $\mathbf{g}_t$ (via a gyroscope in the torso) and foot pressure $\mathbf{f}_t$ (at eight points on the feet) every 1 millisecond. By computing four differential features of the pressure sensors, and extracting the two horizontal gyroscope axis, we form a six dimensional representation of the dynamic state of the robot.

Concatenating the four dimensional kinematic state and the six dimensional dynamic state we form the full ten dimensional state representation $\mathbf{s}_t$. Robot actions $\mathbf{a}_t$ are then simply points in the embedded kinematic space. We bootstrap the forward model (of robot kinematics and dynamics) by first performing random exploration (body babbling) about the instructor's trajectory. Once we have collected sufficient data (around 20 trials) we learn an initial forward model. Subsequently we place a probabilistic constraint on the dynamic configuration of the robot (using a tight, central Gaussian distribution around zero angular velocity, and zero pressure differentials). Using this constraint on dynamics we perform constrained exploration, until we obtain a stable motion for the Hoap-2 which imitates the human motion. The results we obtained in imitating a difficult one-legged balance motion are shown in Fig. 4.

## 5 Conclusion

Our results demonstrate that probabilistic inference and learning techniques can be used to successfully acquire new behaviors in complex robotic systems such as a humanoid robot. In particular, we showed how a nonparametric model of forward dynamics can be learned from constrained exploration and used to infer actions for imitating a teacher while simultaneously taking the imitator's dynamics into account.

There exists a large body of previous work on robotic imitation learning (see, for example [2, 5, 14, 19]). Some rely on producing imitative behaviors using nonlinear dynamical systems (e.g., [8]) while others focus on biologically motivated algorithms (e.g., [3]). In the field of reinforcement learning, techniques such as inverse reinforcement learning [12] and apprenticeship learning [1] have been proposed to learn controllers for complex systems based on observing an expert and learning their reward function. However, the role of this type of expert and that of our human demonstrator must be distinguished. In the former case, the teacher is directly controlling the artificial system. In the imitation learning paradigm, one can only observe the teacher controlling their own body. Further, despite kinematic similarities between the human and humanoid robot, the dynamic properties of the robot and human are very different.

Finally, the fact that our approach is based on inference in graphical models confers two major advantages: (1) we can continue to leverage algorithmic advances in the rapidly developing area of inference in graphical models, and (2) the approach promises generalization to graphical models of more complex systems such as with semi-Markov dynamics and hierarchical systems.

## Footnotes

[1] For detailed algorithms please refer to the technical report available at http://www.cs.washington.edu/homes/grimes/dil

## References

[1] P. Abbeel and A. Y. Ng. Exploration and apprenticeship learning in reinforcement learning. In *In Proceedings of the Twenty-first International Conference on Machine Learning*, 2005.

[2] C. Atkeson and S. Schaal. Robot learning from demonstration. pages 12–20, 1997.

[3] A. Billard and M. Mataric. Learning human arm movements by imitation: Evaluation of a biologically-inspired connectionist architecture. *Robotics and Autonomous Systems*, (941), 2001.

[4] M. A. Carreira-Perpinan. Mode-finding for mixtures of gaussian distributions. *IEEE Trans. Pattern Anal. Mach. Intell.*, 22(11):1318–1323, 2000.

[5] J. Demiris and G. Hayes. A robot controller using learning by imitation, 1994.

[6] D. B. Grimes, R. Chalodhorn, and R. P. N. Rao. Dynamic imitation in a humanoid robot through nonparametric probabilistic inference. In *Proceedings of Robotics: Science and Systems (RSS'06)*, Cambridge, MA, 2006. MIT Press.

[7] A. T. Ihler, E. B. Sudderth, W. T. Freeman, and A. S. Willsky. Efficient multiscale sampling from products of gaussian mixtures. In S. Thrun, L. Saul, and B. Schölkopf, editors, *Advances in Neural Information Processing Systems 16*. MIT Press, Cambridge, MA, 2004.

[8] A. J. Ijspeert, J. Nakanishi, and S. Schaal. Trajectory formation for imitation with nonlinear dynamical systems. In *IEEE/RSJ International Conference on Intelligent Robots and Systems*, pages 752–757, 2001.

[9] F. R. Kschischang, B. J. Frey, and H.-A. Loeliger. Factor graphs and the sum-product algorithm. *IEEE Transactions on Information Theory*, 47(2):498–519, 2001.

[10] W. Li and E. Todorov. Iterative linear-quadratic regulator design for nonlinear biological movement systems. In *Proceedings of the 1st Int. Conf. on Informatics in Control, Automation and Robotics*, volume 1, pages 222–229, 2004.

[11] A. N. Meltzoff. Elements of a developmental theory of imitation. pages 19–41, 2002.

[12] A. Y. Ng and S. Russell. Algorithms for inverse reinforcement learning. In *Proc. 17th International Conf. on Machine Learning*, pages 663–670, 2000.

[13] J. Pearl. *Probabilistic Reasoning in Intelligent Systems: Networks of Plausible Inference*. Morgan Kaufmann, 1988.

[14] S. Schaal, A. Ijspeert, and A. Billard. Computational approaches to motor learning by imitation. 1431:199–218, 2004.

[15] D. Scott and W. Szewczyk. From kernels to mixtures. *Technometrics*, 43(3):323–335.

[16] E. B. Sudderth, A. T. Ihler, W. T. Freeman, and A. S. Willsky. Nonparametric belief propagation. In *CVPR (1)*, pages 605–612, 2003.

[17] H.-G. Sung. *Gaussian Mixture Regression and Classification*. PhD thesis, Rice University, 2004.

[18] Y. Weiss. Correctness of local probability propagation in graphical models with loops. *Neural Computation*, 12(1):1–41, 2000.

[19] M. Y. Kuniyoshi and H. Inoue. "learning by watching: Extracting reusable task knowledge from visual observation of human performance" ieee transaction on robotics and automation, vol.10, no.6, pp.799–822, dec., 1994.
